# Analog LSI Implementation of an Auto-Adaptive Network for Real-Time Separation of Independent Signals

**Marc H. Cohen, Philippe O. Pouliquen and Andreas G. Andreou***
Electrical and Computer Engineering
The Johns Hopkins University, Baltimore, MD 21218, USA

## Abstract

We present experimental data from an analog CMOS LSI chip that implements the Herault-Jutten adaptive neural network. Testing procedures and results in time and frequency-domain are described. These include weight convergence trajectories, extraction of a signal in noise, and separation of statistically complex signals such as speech.

## 1  Introduction

In its most general form, the N x N independent component analyzer (In.C.A.) network (Herault 1986, Jutten 1987, 1991) can be used to solve the following classical signal processing problem; given N physically distinct measurements of *a priori* unknown linear combinations of N independent signal sources, the network auto-adaptively extracts N equivalent independent signals.

The network consists of a set of N simple processing units interconnected by inhibitory synapses (see figure 1). A processing unit $i$ calculates its output $S_i(t)$ based on its input $E_i(t)$ and the weighted sum of the outputs from the remaining $N-1$ units. The weights are updated using a modified Hebbian learning rule (Hebb 1949, Herault 1986, Jutten 1987, 1991).

This architecture has led to various CMOS implementations (Vittoz 1989, Cohen 1991a). We have implemented three different CMOS designs using different learn-

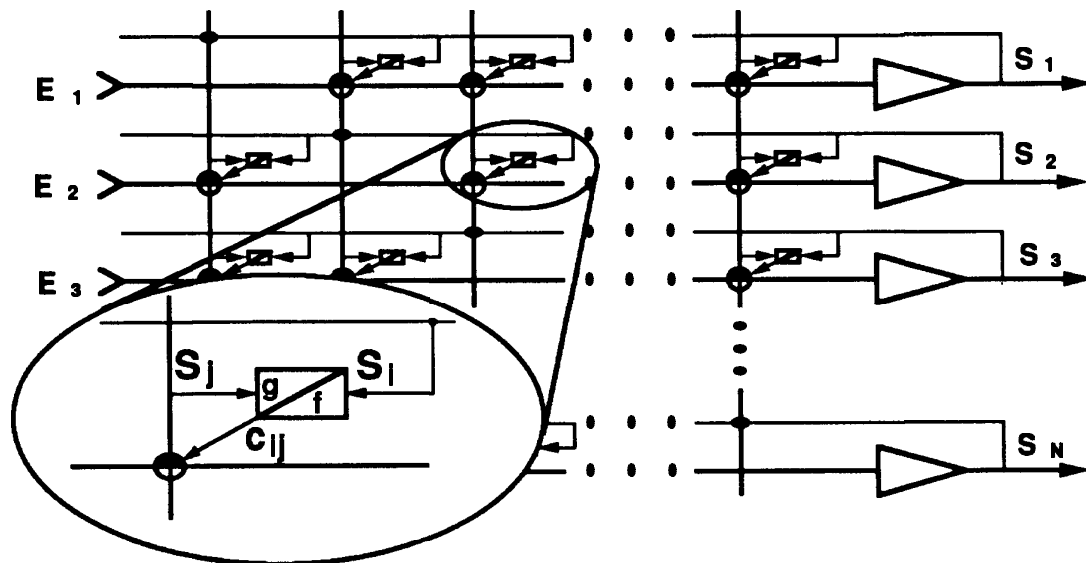

Figure 1: The $N \times N$ network architecture

ing rules, circuits and design methodologies. Two of them employ both above- and below-threshold CMOS circuits, the third (Cohen 1991a, 1992) employs only subthreshold MOS technology. The particulars of the circuits and learning rules employed in our implementations have been described in detail elsewhere (Cohen 1991a, 1991b, 1991c, 1992); this paper concentrates on the test procedures and results using different type of input signals.

In section 2 we describe the test procedure used to observe the evolution of the weights in time from reset to convergence of the network. In section 3 we describe tests designed to observe the frequency domain characteristics of the network. In section 4 we describe more ambitious tests involving speech signals and other audio-band signals.

All results presented here were obtained from the first design: a chip that used the learning rules and design techniques of Vittoz and Arreguit (Vittoz 1989). Our improvements on their original implementation were mostly in the details of the circuits and resulted in a system that had less systematic offsets in the individual components (Cohen 1991a). However, similar tests where performed on the other two designs with similar results.

## 2   Time domain results

This test was chosen to match conditions used for digital simulations of the network, and to compare the evolution of the weights in their weight-space. Two sine waves of approximately 1kHz were mixed in two different ratios, and the mixed signals ($E_1$ and $E_2$) were presented to the chip. The chip output signals ($S_1$ and $S_2$) and the weights ($C_{12}$ and $C_{21}$) were digitized and plotted.

The results are shown in figure 2. Figure 2(a) is a phase plot of the network's input signals, and figure 2(b) is a phase plot of the network's output signals after

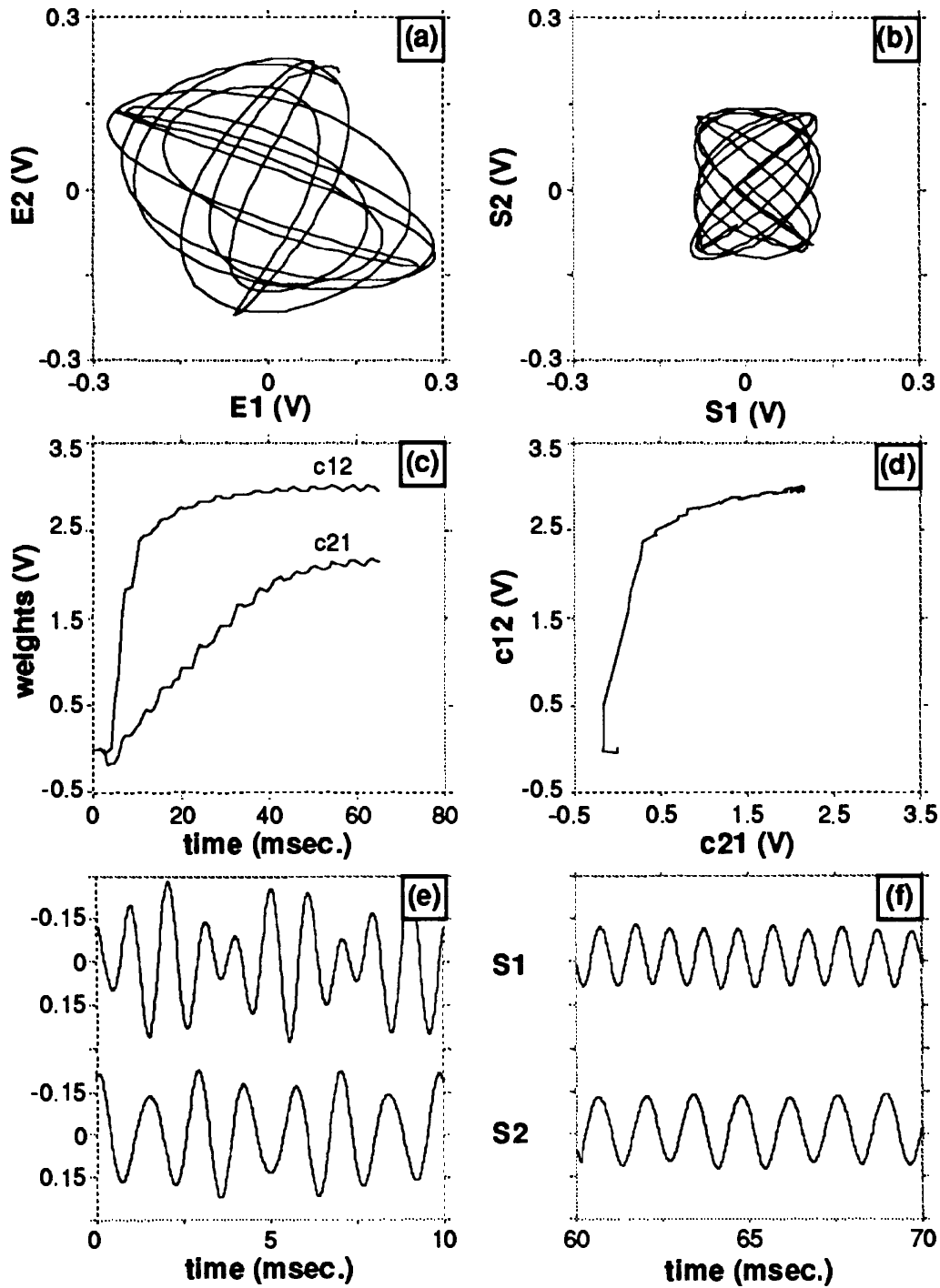

Figure 2: Test results for a 2 × 2 network using two sine waves as input.

convergence of the weights: note how the plot has been transformed from a parallelogram into a rectangle. Figure 2(c) is a plot of the network's weights as a function of time, beginning from the reset state where the weight capacitors are grounded. Figure 2(d) is a phase plot of the network's weights: in this instance, the initial

rapid change in the phase space of the weights is due to offsets in the circuits used. That is, once the system was turned on it assumed an operating point for weights other than the (0,0). Figures 2(e) and 2(f) plot the output signals of the network immediately after reset and after convergence of the weights respectively. Other implementations exhibited similar behaviors except that initial offsets in the weight space (although certainly not catastrophic here) were eliminated by using improved circuit techniques.

This test is not applied to larger networks due to the large number of signals required to observe the weights. By comparing the convergence results of the 2 × 2 networks for which the weights were externally observable with other networks for which the weights were not observable, it was determined that the addition of the observation circuitry slowed convergence by a factor of 5.

# 3   Frequency domain results

This procedure is designed to test an implementation's ability to extract a signal which is "buried" in background noise.

The signal ($X_1$, a sinusoid) is to be extracted from bandpass filtered white noise ($X_2$), which has peak amplitude 10dB greater than the signal and "center" frequency around the frequency of the signal.

Test results are plotted in figure 3: the magnitude spectrum of the original signals, the input signals to the network, and the output signals of the network after convergence are shown. The chip is able to reduce the background noise by 30dB, and extract the sinusoid.

Larger networks (6 × 6) were tested with a mixture of six sinusoidal signals around 1kHz spaced at not regular intervals approximately 20Hz apart. The networks successfully separated each pure sinusoid into a separate output channel and suppressed all adjacent sinusoids by approximately 20dB. No convergence problems were encountered with this larger network.

# 4   Audio-band results

These In.C.A. networks were not necessarily intended for filtering the type of signals that are usually synthesized in a laboratory. Therefore the networks were also tested using music and speech, signals that have more complex statistical properties.

For instance, a recording of a segment of text read in English and a segment of text read in greek by the same speaker were mixed in two slightly different ratios to produce unintelligible input signals for the 2 × 2 networks. The spectrogram of a typical segment of the mixed signals is shown in figure 4. The networks easily recovered the two original recordings: the spectrograms of the outputs are shown

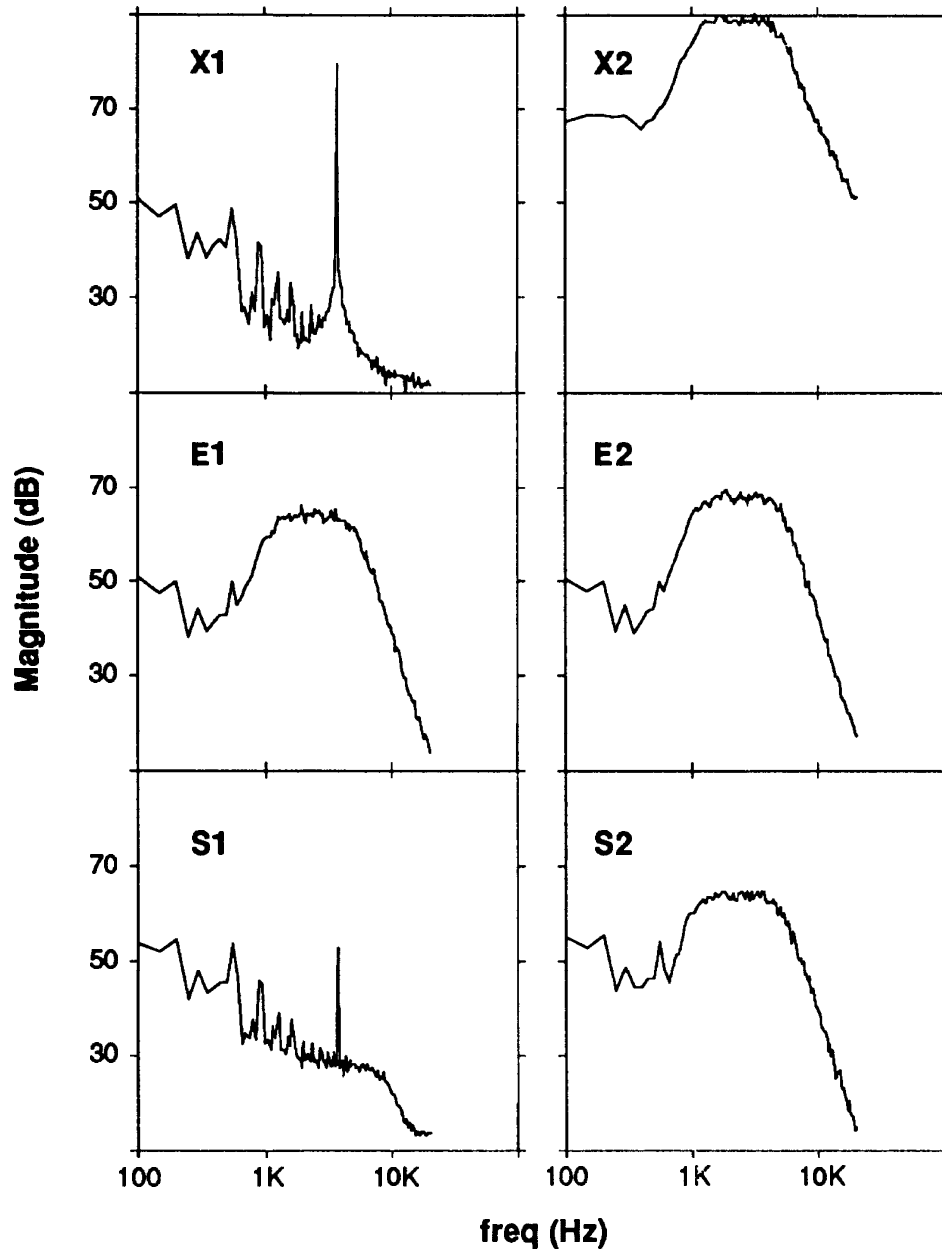

Figure 3: Test results for a 2 × 2 network using a sine wave and bandpass filtered white noise as input.

in figure 5. Similar results were obtained with mixed recordings of music, or combinations of music and speech.

# 5   Conclusion

We have described the results of a network which performs auto-adaptive filtering. By using analog VLSI technology we have achieved a real-time, scalable and low

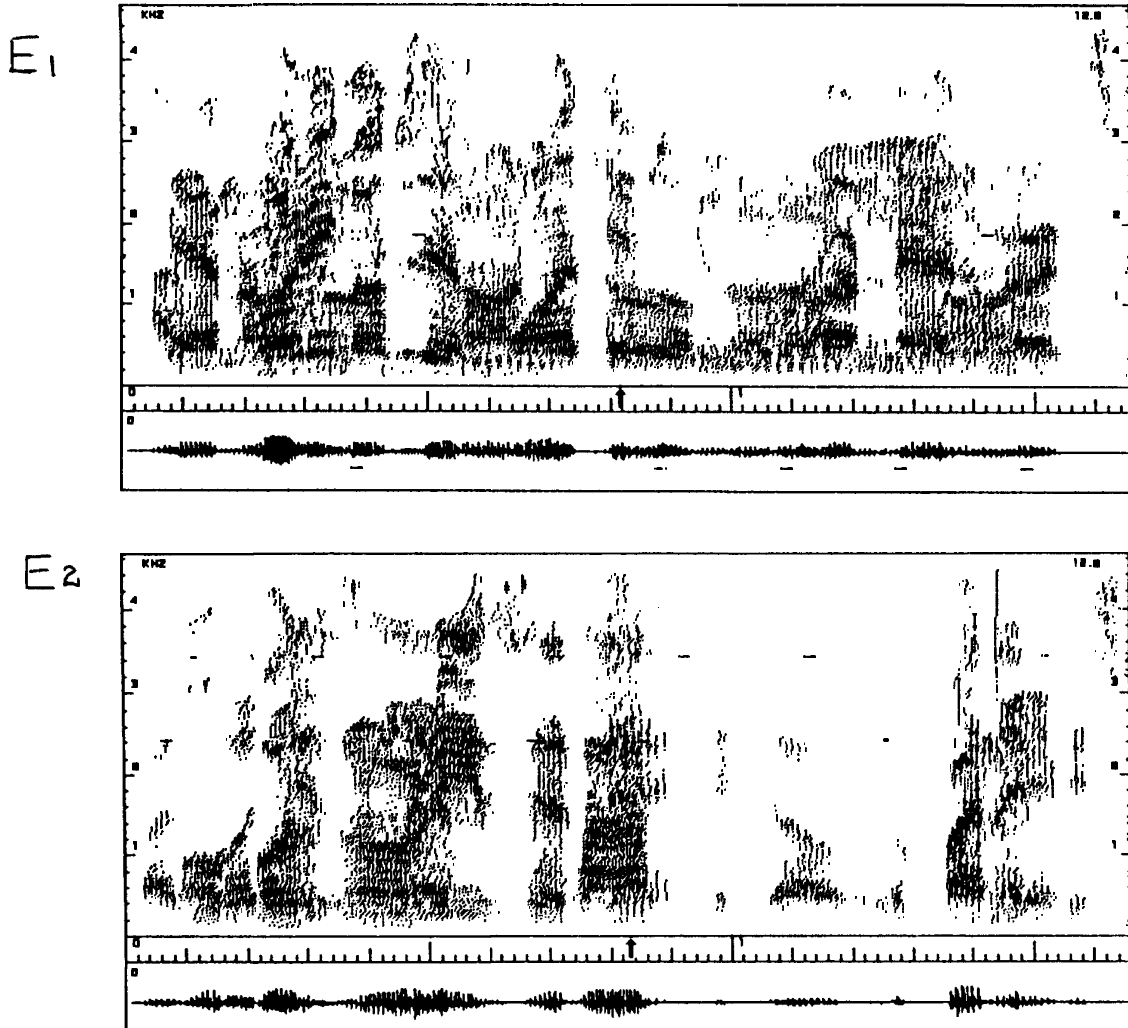

Figure 4: Spectrograms of segments of mixed speech used as input to the 2 × 2 networks.

power realization of the network. We believe it will have many applications, to name but a few;

- three dimensional object reconstruction from stereoscopic vision,

- removing crosstalk in telephone/digital communication lines, and

- separation of evoked potential signals from background EEG and EMG noise.

Using MOS technology and micropower techniques real-time separation of signals in the audio spectrum and up to about 1MHz is possible. Current-mode techniques (Cohen 1992) using bipolar devices, and higher current levels should enable real-time processing of signals of a few hundred MHz.

Future work will involve developing the capability to handle signal delays introduced by the medium through which the signals propagate before reaching the sensors (as

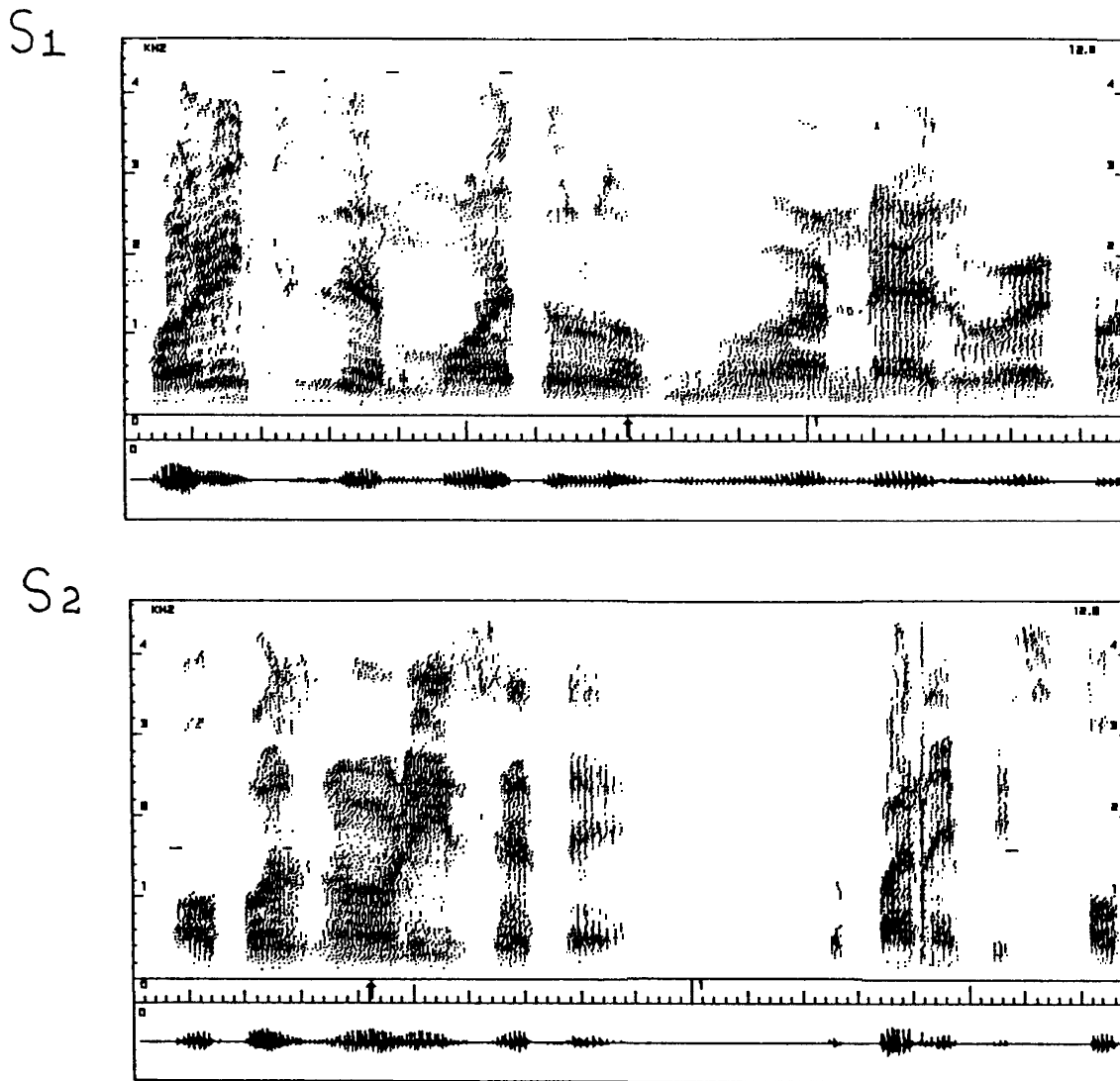

Figure 5: Spectrogram of segments of the output of a 2 x 2 network showing separation of the speech signals.

in the "cocktail party" problem). Such modifications to the algorithm have been proposed by Jutten (Jutten 1987) and recently by Platt and Faggin (Platt 1991).

## Footnotes

*Please address correspondence to Andreas G. Andreou.

## References

Cohen M. H. (1991a) "Analog VLSI Implementation of an Auto-Adaptive Synthetic Neural Network for Real-Time Separation of Independent Signal Sources." M.S.E. Thesis, Biomedical Engineering, The Johns Hopkins University.

Cohen M. H., Pouliquen P. O. and Andreou A. G. (1991b) "Silicon Implementation of an Auto-Adaptive Network for Real-Time Separation of Independent Signals." *Proceedings of the 1991 International Symposium on Circuits and Systems,* Singapore, 2971–2974.

Cohen M. H., Pouliquen P. O. and Andreou A. G. (1991c) "Silicon VLSI Implemen-

tation of an Auto-Adaptive Network for the Real-Time Separation of Independent Signal Sources." *Proceedings of the 25th Annual Conference on Information Sciences and Systems*, The Johns Hopkins University, Baltimore, Maryland, 856–861.

Cohen M. H., Pouliquen P. O. and Andreou A. G. (1991d) "An Auto-Adaptive Synthetic Neural Network for Real-Time Separation of Independent Signal Sources." *Proceedings of the 1991 International Joint Conference on Neural Networks*, Seattle, WA, I-211–214.

Cohen M. H. and Andreou A. G. (1992) "Current-Mode Subthreshold MOS Implementation of the Herault-Jutten Auto-Adaptive Network." *IEEE Journal of Solid-State Circuits* 27(5), May 1992.

Hebb D. O. (1949) *"The Organisation of Behavior"* Wiley, New York.

Herault J. and Jutten C. (1986) "Space or Time Adaptive Signal Processing by Neural Network Models." *Neural Networks for Computing*, AIP Conference Proceedings 151, Snowbird, UT. Edited by John S. Denker.

Jutten C. (1987) "Calcul Neuromimétique et Traitement du Signal, Analyse en Composantes Indépendantes." Ph.D. Thesis, Université Scientifique et Médicale¡ - Institut National Polytechnique, Grenoble, France.

Jutten C. and Herault J. (1991) "Blind Separation of Sources, Part I: An Adaptive Algorithm based on Neuromimetic Architecture." *Signal Processing* 24:1–10, Elsevier Science Publishers.

Platt J. and Faggin F. "A Network for the Separation of Sources that are Superimposed and Delayed." *Advances in Neural Information Processing Systems 4* Morgan Kaufmann Publishers, San Mateo, 1992.

Vittoz E. A. and Arreguit X. (1989) "CMOS Integration of Herault-Jutten Cells for Separation of Sources." *Workshop on Analog VLSI and Neural Systems* Portland, Oregon. Kluwer Academic Press, Norwell, MA.